# A Closer Look at the CLS Token for Cross-Domain Few-Shot Learning

**Yixiong Zou**[1]  **Shuai Yi**[2]  **Yuhua Li**[1]  **Ruixuan Li**[1*]

[1]School of Computer Science and Technology, [2]School of Artificial Intelligence and Automation
Huazhong University of Science and Technology
{yixiongz, yishuai, idcliyuhua, rxli}@hust.edu.cn

## Abstract

Vision Transformer (ViT) has shown great power in learning from large-scale datasets. However, collecting sufficient data for expert knowledge is always difficult. To handle this problem, Cross-Domain Few-Shot Learning (CDFSL) has been proposed to transfer the source-domain knowledge learned from sufficient data to target domains where only scarce data is available. In this paper, we find an intriguing phenomenon neglected by previous works for the CDFSL task based on ViT: leaving the CLS token to random initialization, instead of loading source-domain trained parameters, could consistently improve target-domain performance. We then delve into this phenomenon for an interpretation. We find **the CLS token naturally absorbs domain information** due to the inherent structure of the ViT, which is represented as the low-frequency component in the Fourier frequency space of images. Based on this phenomenon and interpretation, we further propose a method for the CDFSL task to decouple the domain information in the CLS token during the source-domain training, and adapt the CLS token on the target domain for efficient few-shot learning. Extensive experiments on four benchmarks validate our rationale and state-of-the-art performance. Our codes are available at https://github.com/Zoilsen/CLS_Token_CDFSL.

## 1 Introduction

Vision Transformer (ViT) has boosted the development of artificial intelligence due to its great capability in learning from large-scale datasets [3, 10, 11, 18, 26, 39], which is now the foundation model for many deep learning tasks. However, the generalization of ViT on downstream tasks has not been fully explored yet [27, 32, 41], especially under the cross-domain data-scarce scenarios. To handle this problem, Cross-Domain Few-Shot Learning (CDFSL) [14, 21, 28, 33, 47] has been proposed to transfer the general knowledge from the data-sufficient source domain (such as ImageNet [8]) to learn the expert knowledge in the data-scarce target domain (such as medical datasets [38]). Due to the domain gap and scarce training data, CDFSL still remains a challenging task.

To handle this task, we focus on the basic transfer-learning-based method, i.e., training the ViT model on the source domain by the all-class classification loss, and loading the model parameters on the target domain for classification. We find an intriguing phenomenon that is ignored by previous works: on the target domain dataset, not loading the CLS token parameters (i.e., leaving the CLS token to random initialization) would consistently lead to higher performance in most cases, although such operation may harm the source-domain performance, as shown in Fig. 1.

In this paper, we delve into this phenomenon for an interpretation. We first study what information is encoded by the CLS token during the source-domain training. We find **the CLS token naturally**

---

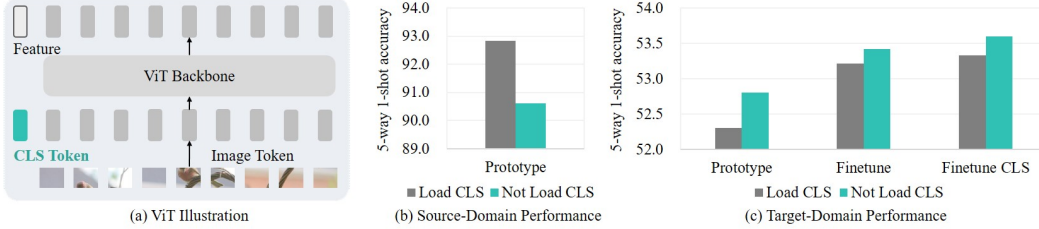

(a) ViT Illustration     (b) Source-Domain Performance     (c) Target-Domain Performance

Figure 1: (a) Vision Transformer (ViT) takes image tokens and a learnable CLS token as input. For the Cross-Domain Few-Shot Learning (CDFSL) task, after the training on the source-domain dataset, we evaluate the model on both the source-domain classes (b) and target-domain classes (c) by the 5-way 1-shot classification. We find an intriguing phenomenon neglected by previous works: although not loading the CLS token parameters (i.e., leaving them to random initialization) on the source-domain classes harms the performance (b), **not loading these parameters consistently improves the target-domain performance (c)**. In this paper, we delve into this phenomenon for an interpretation, and propose a simple but effective method based on them for the CDFSL task.

**absorbs domain information**, represented as low-frequency components in the Fourier frequency space of images, due to the ViT's inherent structure which places the CLS token at the input layer and shares it with all input image patches. Based on this phenomenon and interpretation, we further propose a method for the CDFSL task. During the source-domain training, this method decouples the domain information from the CLS token to make it domain agnostic, handling the domain gap problem. During the target-domain learning, this method finetunes the CLS token to efficiently absorb domain information, handling the few-shot learning problem. Experiments on four benchmark datasets validate our rationale, and show that we can outperform current state-of-the-art works.

In summary, our contributions can be listed as follows.

- We find a phenomenon that is neglected by previous works: not loading the source-domain trained CLS token could consistently improve the performance for the target-domain generalization.

- We delve into this phenomenon for an interpretation: the CLS token naturally absorbs domain information due to the inherent structure of vision transformers.

- Based on this interpretation, we further propose a method to decouple the domain information in the CLS token, and make use of the CLS token for efficient downstream few-shot learning.

- Extensive experiments on four CDFSL benchmark datasets validate the rationale of our interpretation and method, showing we can outperform current state-of-the-art works.

## 2   Delve into the CLS Token for Cross-Domain Few-Shot Learning

### 2.1   Preliminaries

Cross-domain few-shot learning (CDFSL) requires the model to be firstly trained on a source-domain dataset (e.g., *mini*ImageNet [34]), then applied to learn from a few training samples from each target-domain dataset, and finally evaluated on the test set of the target dataset. Specifically, denoting the source dataset as $D^S = \{x_i^S, y_i^S\}_{i=1}^N$, the baseline method is to train the model $f(\cdot)$ on $D^S$ by minimizing the cross-entropy loss against the source-domain label space $|Y^S|$ as

$$L_{CE} = \frac{1}{N} \sum_i^N CE(f(x_i^S), y_i^S). \tag{1}$$

During the learning on each target dataset, the model will be applied to learn from the sampled support set $\{x_{ij}^T, y_{ij}^T\}_{i=1,j=1}^{K,M}$ which is called the $K$-way $M$-shot task (i.e., $K$ classes in each support set with $M$ samples in each class). Finally, the model will be evaluated on the query set $\{x_q^T\}$. Typically, the classification is based on the distance between class prototypes and the test samples as

$$\hat{y_q^T} = \arg\min_i d(\frac{1}{M} \sum_j f(x_{ij}^T), f(x_q^T)), \tag{2}$$

Table 1: Not Loading the CLS token improves the cross-domain performance in most cases.

| Method | CropDisease | EuroSAT | ISIC2018 | ChestX | Ave. |
|---|---|---|---|---|---|
| (a.1) Baseline | 81.22 | 72.08 | 32.75 | 22.84 | 52.22 |
| (a.2) Baseline + Fix CLS as Initialization | 82.50 +1.28 | 72.58 +0.50 | 32.69 -0.06 | 22.94 +0.10 | 52.68 +0.46 |
| (a.3) Baseline + not loading CLS | 82.54 +1.32 | 73.02 +0.94 | 32.76 +0.01 | 22.84 +0.06 | 52.81 +0.59 |
| (b.1) Baseline + finetuning FC | 83.14 | 72.81 | 33.82 | 23.09 | 53.22 |
| (b.2) Baseline + finetuning FC + not loading CLS | 83.32 +0.18 | 73.80 +0.99 | 33.37 -0.45 | 23.19 +0.10 | 53.42 +0.21 |
| (c.1) Baseline + finetuning FC & CLS | 83.15 | 73.26 | 33.84 | 23.07 | 53.33 |
| (c.2) Baseline + finetuning FC & CLS + not loading CLS | 83.37 +0.22 | 73.63 +0.37 | 34.26 +0.42 | 23.13 +0.06 | 53.60 +0.27 |

where $f(x) \in R^d$ is the feature, and $d(\cdot, \cdot)$ denotes the Euclidean distance. Repeating the above sampling-evaluation process multiple times on target datasets, the performance will be obtained.

In this paper, we set the model $f(\cdot)$ to the Vision Transformer (ViT) [10] with DINO [42] (other settings can be found in the appendix). ViT takes both the image tokens and a learnable CLS token as inputs, written as

$$f(x) = h(T^C, T(x)) \tag{3}$$

where $T^C \in R^d$ is the CLS token, $T(x) \in R^{n^t \times d}$ is the image tokens, and $h(\cdot, \cdot)$ processes the combination of two kinds of tokens, as shown in Fig. 1a.

## 2.2 Wider verification of the phenomenon

To delve into this phenomenon, we first study this phenomenon under different training conditions for a detailed verification. Following [13], we first take the ViT model pretrained on the ImageNet dataset as an initialization. Then, the model is trained on the base classes of the *mini*ImageNet dataset as the source-domain training. Finally, we learn and evaluate the model on four benchmark datasets (CropDiseases [25], EuroSAT [16], ISIC2018 [5] and ChestX [38]) by the 5-way 1-shot accuracy. The target-domain learning methods include prototype-based method (Tab. 1a), which does not need to adapt the model parameters; finetuning-based methods (Tab. 1b,c), which only adapts parts of model parameters due to the scarcity of training data. Results are reported in Tab. 1, where both the accuracy and the change of accuracy w.r.t. the first row of each block are included.

From this table, we can see **this phenomenon exists in most cases**, no matter whether we finetune the model or not. This means the CLS token trained on the source domain may contain poisonous information for target-domain generalization. Therefore, we also try to finetune the CLS token (Tab. 1c) on the target domain, but this phenomenon still exists, which means **such source-domain information is strong enough that it cannot be simply handled by the target-domain finetuning**.

To ablate such information from the source-domain training, we directly fix the CLS token as random initialization for both the source and target domains (Tab. 1 a.2). We can see that by abandoning the learning of the CLS token, the performance is also improved from the baseline method, but is slightly lower than training but not loading it (Tab. 1 a.3). This means **such information in the CLS token could be beneficial for the source-domain training**. Therefore, in the following subsections, we will keep delving into such information to study what it is and how it is learned.

## 2.3 What information does the CLS token encode?

Intuitively, since not loading the CLS token improves performances only under cross-domain scenarios, it is natural to think of the CLS token's poisonous information as the domain information. Therefore, below we will validate and delve into this intuition both quantitatively and qualitatively.

### 2.3.1 Quantitative study: CLS token contains domain information

Firstly, we quantitatively measure the domain distance between source and target datasets by the CKA similarity following [7]. Specifically, given a backbone network, we extract features from images in different domains, and then calculate the CKA similarity by aligning the channel dimension. The larger the CKA similarity is, the smaller the domain distance will be, and it means the model contains less domain information. The results are plotted in Fig. 2a. We can see:

(1) Not loading the CLS token can significantly increase the CKA similarity, indicating the CLS token contains domain information while other structures tend to capture domain-irrelevant information.

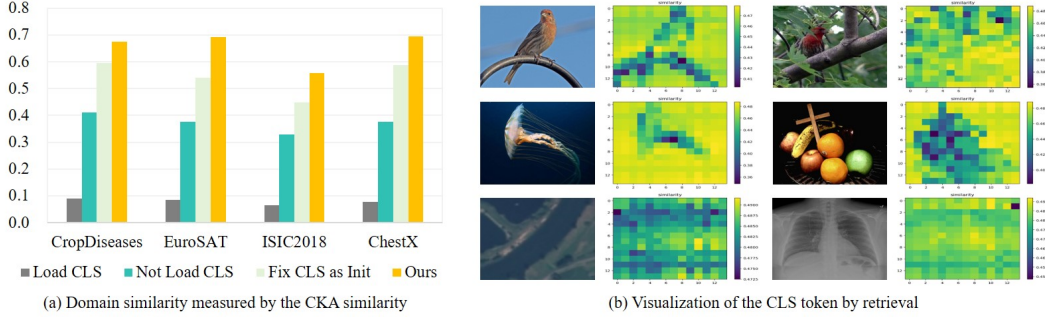

(a) Domain similarity measured by the CKA similarity

(b) Visualization of the CLS token by retrieval

Figure 2: (a) Not loading the CLS token significantly improves the domain similarity, indicating the CLS token contains domain information. (b) The similarity map between the CLS token and image tokens can roughly represent the background of the object (top two rows), which can hardly be transferred to target domains (bottom row). However, in some images (e.g., first row, second column), the highlighted regions are not necessarily the background but the dim regions (bottom-right region), which inspires us to consider whether the CLS token actually captures the low-frequency components in the Fourier frequency space of images.

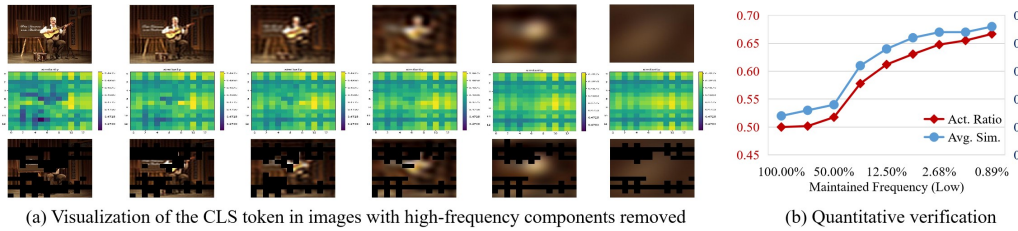

(a) Visualization of the CLS token in images with high-frequency components removed

(b) Quantitative verification

Figure 3: (a) We only maintain the low-frequency component of images, and the frequency threshold decreases from left to right. The similarity map becomes brighter with the decreasing of the frequency, indicating the CLS token shows higher similarity to low-frequency components. (b) We quantitatively measure the average value of the similarity map and ratio of activated regions for different low-frequency images. With the decrease of the threshold, the similarity consistently increases.

(2) Fixing the CLS token as random initialization in the source-domain training can further increase the CKA similarity, because the domain information inherited from the pertaining is now limited.

Therefore, we validate that the CLS token indeed contains domain information, and it plays an important role in learning such information.

### 2.3.2  Qualitative study: CLS token captures low Fourier-frequency components

To study the information encoded by the CLS token, we then visualize the CLS token by image retrieval. Specifically, we calculate the cosine similarity between the CLS token and the image token in the input layer, and report the similarity map in Fig. 2b. We can see the similarity map could roughly represent the contour of the object for the source dataset, by **highlighting the background regions with higher similarity**. Since identifying the background also means the model could find the foreground in the image, the CLS token can indeed facilitate the source-domain recognition. However, on the target datasets, such capability is downgraded to activating just random regions (EuroSAT, ChestX), which limits the effectiveness of the CLS token and is consistent with the performance change in Tab. 1.

Since the background region in ImageNet is always in bokeh due to the photography tools, we observe these regions as being blurry. This inspires us to consider whether the CLS token actually tends to find the low-frequency regions (with the Fourier Transform). To validate this hypothesis, we try to remove the high-frequency components and maintain only the low-frequency components of input images, and compare the generated similarity map in Fig. 3a. Intuitively, by removing the frequency components from high to low (from left to right in Fig. 3a), the input image becomes blurrier. The corresponding similarity map, as the brighter color indicates a higher similarity, gradually gets brighter from left to right. This indicates **the information encoded by the CLS token is more likely to be contained by these blurry regions**. Quantitatively, we measure the average similarity value of

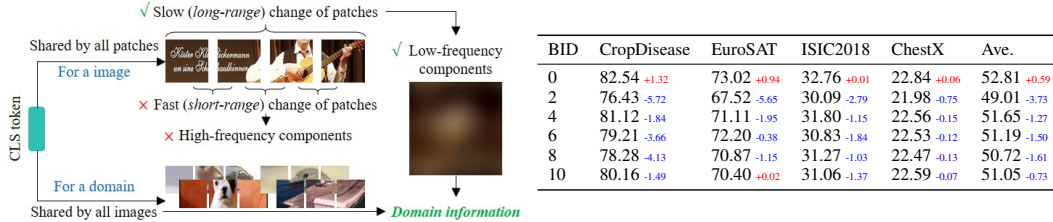

| BID | CropDisease | EuroSAT | ISIC2018 | ChestX | Ave. |
|---|---|---|---|---|---|
| 0 | 82.54 +1.32 | 73.02 +0.94 | 32.76 +0.01 | 22.84 +0.06 | 52.81 +0.59 |
| 2 | 76.43 -5.72 | 67.52 -5.65 | 30.09 -2.79 | 21.98 -0.75 | 49.01 -3.73 |
| 4 | 81.12 -1.84 | 71.11 -1.95 | 31.80 -1.15 | 22.56 -0.15 | 51.65 -1.27 |
| 6 | 79.21 -3.66 | 72.20 -0.38 | 30.83 -1.84 | 22.53 -0.12 | 51.19 -1.50 |
| 8 | 78.28 -4.13 | 70.87 -1.15 | 31.27 -1.03 | 22.47 -0.13 | 50.72 -1.61 |
| 10 | 80.16 -1.49 | 70.40 +0.02 | 31.06 -1.37 | 22.59 -0.07 | 51.05 -0.73 |

Table 2: (Left) Interpretation of why the CLS token captures domain information. (Right) Only not loading the CLS token in the first block can consistently improve performance.

*all images* in Fig. 3b as the blue curve to verify such intuition. We can see such values consistently increase with the decrease of maintained frequency.

Moreover, we then utilize the similarity map to generate the segmentation mask of the input image in the third row, where we set the masking threshold as the 50%-th similarity value in the original similarity map. Intuitively, we can observe that (1) with the decrease of the frequency, the masked regions get fewer, and (2) the regions (the guitarist, foreground) that are originally masked in the all-frequency (left-most) image are not masked in the low-frequency (right-most) image. This indicates the CLS token actually does not detect the fore- or background, because if the foreground (the guitarist) gets into low-frequency regions, the CLS token can no longer detect it, which verifies that **it is the low-frequency components that the CLS token detects**. Quantitatively, we measure the ratio of regions that surpass the threshold (Act. Ratio) in Fig. 3b as the red curve to verify such intuition. We can also see such ratios consistently increase as the frequency gets lower in all images[2].

As the low-frequency information has been verified to be relevant to the domain information [12], the above experiments also validate that **the CLS token encodes the domain information**.

In all, by qualitative and quantitative experiments, we verify the CLS token encodes domain information, which contributes to the source-domain training but is harmful to target-domain generalization.

### 2.4 Why does the CLS token contain domain information?

Finally, we interpret why the CLS token encodes the domain information in two folds (Tab. 2):

**(1) The CLS token is shared by all image-patch tokens**. As shown in Tab. 2 (left), the influence of such characteristics can be divided into two folds: (a) For each image, the CLS token adapts much slower than image tokens, so it cannot reflect the patch-wise change in each image. This means it tends to capture the trend of a relatively large number of patches, which is reflected in low-frequency components in the frequency space and is verified to be relevant to domain information. (b) For each dataset, the CLS token captures the shared information for all patches. What information is shared by all patches in each dataset: the domain information, coinciding with the influence for each image.

**(2) The CLS token captures low-level information** that is more vulnerable to domain information. To verify this hypothesis, we move the CLS token to different ViT blocks and train the model on the source dataset, and report the performance change by loading the CLS token or not in Tab. 2 (right). We can see only the first block (i.e., the input layer which is the default choice in ViT) shows consistent performance improvements by not loading the CLS token, which verifies our hypothesis.

### 2.5 Conclusion and Discussion

Based on the above experiments, we make the interpretation as follows. The CLS token is originally designed to be placed at the input layer, which is treated as a shared token for different input tokens. These two characteristics enable the CLS token to naturally absorb the domain information, since (1) the low-level information in the input layer is vulnerable to domain changes, and (2) the CLS token is shared by all image tokens therefore it captures the low-frequency information in each image that is relevant to domain changes. Such information is beneficial for source-domain learning as it partly highlights the background, but can hardly be transferred to target domains. As a result, not loading the CLS token harms the source-domain performance but improves the target-domain performance.

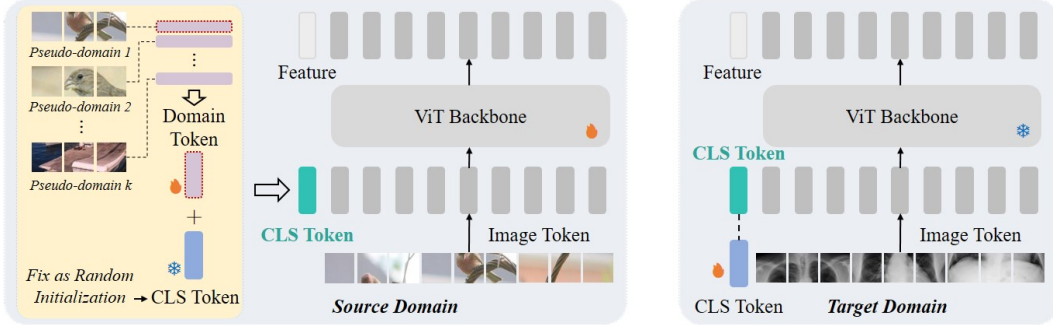

Figure 4: Based on the phenomenon and interpretation, we propose to decouple the domain information from the CLS token to make it domain-agnostic during the source-domain training, and utilize the CLS token's characteristic in absorbing domain information for efficient target-domain adaptation. Specifically, during the source-domain training, we generate pseudo domains on the source dataset by clustering, and apply a domain token for each pseudo domain. We fix the CLS token as the random initialization, and add the domain token to the fixed CLS token, so that **domain tokens will substitute the CLS token in absorbing domain information**, which decouples the domain information from the CLS token. During the target-domain adaptation, we **abandon domain tokens and finetune the CLS token to absorb target-domain information** for efficient few-shot learning.

Moreover, we draw inspirations for the model design to facilitate the target-domain generalization and few-shot learning: (1) The model should on the one hand maintain the learning of CLS token due to its benefits in source-domain learning, and on the other hand alleviate its negative effect in domain bias. Simply not loading the CLS token on the target domain cannot fully take advantage of the CLS token's characteristics. (2) The CLS token is beneficial for downstream few-shot learning, as it efficiently encodes domain information with just a few parameters. Therefore, we can focus on the finetuning of the CLS token for the downstream fast adaptation (akin to the prompt tuning [15, 31, 37, 40]).

## 3 Method: Decoupling Domain Information from the CLS Token

Based on the above interpretation, we aim to make use of the CLS token's characteristic of absorbing domain information for the CDFSL task. Therefore, we propose to decouple the domain information naturally absorbed by the CLS token during the source-domain training to make the CLS token domain-agnostic, and finetune the CLS token for efficient target-domain few-shot learning (Fig. 4).

For the source-domain stage, our decoupling method includes three modifications to Sec. 2.1:

(1) Not loading the pre-trained CLS token parameters, since the domain information absorbed in the CLS token does not help the CDFSL task.

(2) Random initialize the CLS token, so that it does not contain any domain information, and then fix it, as the learning of the CLS token will naturally absorb domain information.

(3) Explicitly adding source-domain-specific tokens to the CLS token, which would **substitute the CLS token to absorb the domain information** and will not be applied on the target domain.

In summary, we aim to decouple the original CLS token into a source-domain-specific part (we call domain tokens) and a source-domain-agnostic part (the fixed CLS token), where the domain tokens will be abandoned on target domains, and the CLS token fixed on source domains will be finetuned on target domains for efficient few-shot learning. Therefore, we need to encourage domain tokens to absorb domain information more effectively. However, although the original CLS token could naturally absorb domain information, its efficacy may not be optimal, since the original CLS token is shared by all images in the source dataset, but the source dataset could demonstrate sub-domains in it.

To handle this problem, we create sub-domains by clustering the class prototypes (i.e., fully-connected layer parameters) generated by the baseline model. For clarity, we call these created domains as pseudo domains. Then, we assign each pseudo domain with an individual domain token, which makes such domain token only absorb information specific to such pseudo domain, *by means of sharing different domain tokens among different pseudo-domain images, instead of sharing one domain token*

*among all source-domain images*. This improves the efficacy of absorbing domain information, thereby helping source-domain learning.

Specifically, following section 2.1, we denote the ViT backbone network as $f(x) = h(T^C, T(x))$ where $T^C \in R^d$ is the CLS token and $T(x) \in R^{n^t \times d}$ is image tokens. Then, we manually specify a set of learnable domain tokens $T^D \in R^{n^d \times d}$ to represent $n^d$ pseudo domains on the source dataset. Given an input image $x$, besides its source-domain label $y$, we also have access to its pseudo-domain label $z$, since these pseudo domains are created by ourselves. Therefore, we can obtain the domain token for this sample as $T_z^D \in R^d$. Subsequently, we replace the original input CLS token $T^C$ by the combination of the domain token and the CLS token as

$$f(x) = h(T^C + T_z^D, T(x)).  \tag{4}$$

Ideally, since now $T^C$ is fixed as the random initialization, $T_z^D$ would substitute $T^C$ to learn the information that is originally captured by $T^C$. Since we explicitly specify different $T_z^D$ for each pseudo domain, the domain information will be better absorbed by the domain tokens.

To explicitly encourage the decoupling of domain information, we also apply another loss to make the domain token orthogonal to the CLS token, which is represented as

$$L_{orth} = \frac{1}{n^t} \sum_z^{n^t} |\frac{T^C}{||T^C||} \frac{T_z^D}{||T_z^D||}|  \tag{5}$$

Finally, the model is trained by the cross-entropy loss in Eq. 1 and $L_{orth}$ with a hyper-parameter $\lambda$ as

$$L = L_{CE} + \lambda L_{orth}.  \tag{6}$$

For the target-domain stage, we abandon the domain tokens as they capture the source-domain information, and directly utilize the CLS token as Eq. 3. For the prototype-based classification, we follow Eq. 2 for target-domain recognition. For the finetuning-based classification, note that there is no domain gap between the training set and the test set on the target domain. Since the CLS token naturally absorbs domain information, now we set the CLS token to learnable parameters to encourage it to absorb target-domain information. The finetuning is based on the cross-entropy loss on the support set, and we only train the classifier and the CLS token for efficient adaptation.

## 4 Experiments

### 4.1 Dataset and Implementation Details

Following current works [28, 33], we utilize the *mini*ImageNet dataset [34] as our source domain with around 60k images and 100 annotated classes. We train our models on the training split of the source dataset, then finetune and evaluate the generalization performance on four target-domain datasets, CropDisease [25], EuroSAT [16], ISIC [5], and ChestX [38], which are cross-domain datasets from the domain of agriculture, remote sensing and medical data (with significant domain gaps).

In implementation, we set the domain number to 64, i.e., each source-domain class has a specific domain token. We set $\lambda$ to 100 to keep two losses on the same scale. We follow [1, 13, 42] to take DINO on ImageNet as the pretraining of our backbone network, and scale the learning rate of the domain token to 1% of the backbone network. Experiments are conducted on NVIDIA A5000 GPUs.

### 4.2 Comparison with State-of-the-Art Works

We report our comparison with state-of-the-art works utilizing the ViT-S backbone for both 5-shot and 1-shot settings in Tab. 3 and 4. For a fair comparison, we compare works with and without finetuning (FT) respectively. The asterisk (*) denotes a transductive setting. As can be seen, our results achieve the top average performance in all settings and consistently outperform current works, MEM-FS [35], StyleAdv [13], PMF [29] and FLoR [46], in almost all datasets.

### 4.3 Ablation Study

The ablation study is reported in Tab. 5 for the prototype-based classification which studies the source-domain training. We can see both the CLS decoupling and the orthogonal loss contribute to

Table 3: Comparison with state-of-the-art works by the 5-way 5-shot classification.

| Methods | backbone | FT | Mark | Crop. | Euro. | ISIC. | Ches. | Ave. |
|---|---|---|---|---|---|---|---|---|
| LDP-net [44] | ResNet10 | × | CVPR-23 | 89.40 | 82.01 | 48.06 | 26.67 | 61.29 |
| GNN+AFA [17] | ResNet10 | × | ECCV-22 | 88.06 | 85.58 | 46.01 | 25.02 | 61.67 |
| SDT [22] | ResNet10 | × | NN-24 | 90.27 | 82.02 | 48.66 | 27.20 | 62.04 |
| FLoR [46] | ResNet10 | × | CVPR-24 | 91.25 | 80.87 | **51.44** | 26.70 | 62.32 |
| MEM-FS [35] | ViT-S | × | TIP-23 | 93.74 | 86.49 | 47.38 | 26.67 | 63.57 |
| StyleAdv [13] | ViT-S | × | CVPR-23 | 94.85 | 88.57 | 47.73 | 26.97 | 64.53 |
| MICM [43] | ViT-S | × | MM-24 | 94.61 | 90.08 | 46.85 | 27.11 | 64.66 |
| SDT [22] | ViT-S | × | NN-24 | 95.00 | 89.60 | 47.64 | 26.72 | 64.75 |
| FLoR [46] | ViT-S | × | CVPR-24 | 95.28 | 90.41 | 49.52 | 26.71 | 65.48 |
| **CD-CLS** | ViT-S | × | **Ours** | **95.68** | **91.04** | 50.46 | **27.23** | **66.10** |
| FLoR [46] | ResNet10 | ✓ | CVPR-24 | 92.33 | 83.06 | **56.74** | 26.77 | 64.73 |
| PMF [29] | ViT-S | ✓ | CVPR-22 | 92.96 | 85.98 | 50.12 | 27.27 | 64.08 |
| StyleAdv [13] | ViT-S | ✓ | CVPR-23 | 95.99 | 90.12 | 51.23 | 26.97 | 66.08 |
| FLoR [46] | ViT-S | ✓ | CVPR-24 | **96.47** | 90.75 | 53.06 | 27.02 | 66.83 |
| **CD-CLS** | ViT-S | ✓ | **Ours** | 96.27 | **91.53** | 54.69 | **27.66** | **67.54** |
| LDP-net* [44] | ResNet10 | ✓ | CVPR-23 | 91.89 | 84.05 | 48.44 | 26.88 | 62.82 |
| RDC* [19] | ResNet10 | ✓ | CVPR-22 | 93.30 | 84.29 | 49.91 | 25.07 | 63.14 |
| FLoR* [46] | ResNet10 | ✓ | CVPR-24 | 93.60 | 83.76 | **57.54** | 26.89 | 65.45 |
| MEM-FS+RDA* [35] | ViT-S | ✓ | TIP-23 | 95.04 | 88.77 | 51.02 | 27.98 | 65.70 |
| **CD-CLS*** | ViT-S | ✓ | **Ours** | **96.62** | **91.68** | 55.66 | **28.25** | **68.05** |

Table 4: Comparison with state-of-the-art works by the 5-way 1-shot classification.

| Method | backbone | FT | Mark | Crop. | Euro. | ISIC. | Ches. | Ave. |
|---|---|---|---|---|---|---|---|---|
| GNN+AFA [17] | ResNet10 | × | ECCV-22 | 67.61 | 63.12 | 33.21 | 22.92 | 46.97 |
| LDP-net [44] | ResNet10 | × | CVPR-23 | 69.64 | 65.11 | 33.97 | 23.01 | 47.18 |
| FLoR [46] | ResNet10 | × | CVPR-24 | 73.64 | 62.90 | **38.11** | 23.11 | 49.69 |
| SDT [22] | ResNet10 | × | NN-24 | 73.92 | 65.87 | 36.45 | **23.22** | 49.97 |
| MEM-FS [35] | ViT-S | × | TIP-23 | 81.11 | 68.11 | 32.97 | 22.76 | 51.24 |
| StyleAdv [13] | ViT-S | × | CVPR-23 | 81.22 | 72.15 | 33.05 | 22.92 | 52.34 |
| SDT [22] | ViT-S | × | NN-24 | 81.03 | 72.71 | 33.40 | 22.79 | 52.48 |
| FLoR [46] | ViT-S | × | CVPR-24 | 81.81 | 72.39 | 34.20 | 22.78 | 52.80 |
| **CD-CLS** | ViT-S | × | **Ours** | **83.51** | **74.08** | 34.21 | 22.93 | **53.68** |
| FLoR [46] | ResNet10 | ✓ | CVPR-24 | 84.04 | 69.13 | **38.81** | 23.12 | 53.78 |
| PMF [29] | ViT-S | ✓ | CVPR-22 | 80.79 | 70.74 | 30.36 | 21.73 | 50.91 |
| FLoR [46] | ViT-S | ✓ | CVPR-24 | 83.55 | 73.09 | 35.49 | 23.26 | 53.85 |
| StyleAdv [13] | ViT-S | ✓ | CVPR-23 | 84.11 | 74.93 | 33.99 | 22.92 | 53.99 |
| **CD-CLS** | ViT-S | ✓ | **Ours** | **84.54** | **74.97** | 35.56 | **23.39** | **54.62** |
| LDP-net* [44] | ResNet10 | ✓ | CVPR-23 | 81.24 | 73.25 | 33.44 | 22.21 | 52.54 |
| RDC* [19] | ResNet10 | ✓ | CVPR-22 | 85.79 | 70.51 | 36.28 | 22.32 | 53.73 |
| FLoR* [46] | ResNet10 | ✓ | CVPR-24 | 86.30 | 71.38 | **41.67** | 23.12 | 55.62 |
| MEM-FS+RDA* [35] | ViT-S | ✓ | TIP-23 | 83.74 | 75.91 | 37.07 | 23.85 | 55.14 |
| **CD-CLS*** | ViT-S | ✓ | **Ours** | **87.39** | **78.41** | 37.20 | **23.88** | **56.72** |

the performance, and the CLS decoupling contributes the most. Moreover, we conduct experiments on the second block to compare different design choices from our final method.

### 4.3.1 Verification of fixing CLS token as random initialization

Since our method fixes the CLS token as the random initialization during the source-domain training, we first try to load the DINO pretrained CLS token (Tab. 5a). The performance drops from 66.10 to 64.46. This is because the CLS token's capture of domain information is not bound to the label supervision. Although DINO is a self-supervised learning method, it still absorbs strong domain information from ImageNet, therefore we can hardly decouple the domain information from it.

Then, we try to set the CLS token to learnable parameters (Tab. 5b), and we can see the performance still drops from 66.10 to 64.83, which is slightly higher than (Tab. 5a). This is because although we explicitly set the domain token for decoupling, the characteristic of the CLS token does not change: it is still located in the input layer and changes slowly with different input tokens. Therefore, it still captures the domain information as long as we set it to learnable parameters, although the captured domain information is not as strong as that from DINO.

Table 5: Ablation study of source-domain training by the 5-way 5-shot accuracy.

| Method | CropDisease | EuroSAT | ISIC2018 | ChestX | Ave. |
|---|---|---|---|---|---|
| Baseline | $94.62_{\pm0.26}$ | $88.62_{\pm0.22}$ | $46.08_{\pm0.33}$ | $26.25_{\pm0.17}$ | $63.89_{\pm0.14}$ |
| + Decoupling | $95.55_{\pm0.22}$ | $90.48_{\pm0.25}$ | $49.58_{\pm0.26}$ | $27.03_{\pm0.17}$ | $65.66_{\pm0.15}$ |
| + Orth | $\mathbf{95.68}_{\pm0.21}$ | $\mathbf{91.04}_{\pm0.29}$ | $\mathbf{50.46}_{\pm0.19}$ | $\mathbf{27.23}_{\pm0.18}$ | $\mathbf{66.10}_{\pm0.17}$ |
| (a) Load CLS | $94.93_{\pm0.21}$ | $88.99_{\pm0.28}$ | $47.15_{\pm0.23}$ | $26.77_{\pm0.16}$ | $64.46_{\pm0.14}$ |
| (b) Learn CLS | $95.36_{\pm0.22}$ | $89.79_{\pm0.25}$ | $47.12_{\pm0.22}$ | $26.94_{\pm0.20}$ | $64.83_{\pm0.15}$ |
| (c) w/o Domain Token | $94.84_{\pm0.26}$ | $89.54_{\pm0.30}$ | $46.63_{\pm0.25}$ | $26.99_{\pm0.19}$ | $64.50_{\pm0.16}$ |
| (d) Not learn Domain Token | $95.40_{\pm0.27}$ | $90.67_{\pm0.26}$ | $48.45_{\pm0.24}$ | $27.09_{\pm0.16}$ | $65.37_{\pm0.15}$ |
| (e) Random Domain Choice | $94.76_{\pm0.23}$ | $89.87_{\pm0.22}$ | $46.08_{\pm0.21}$ | $26.82_{\pm0.17}$ | $64.38_{\pm0.15}$ |
| (f) Decouple by Prompt | $95.07_{\pm0.22}$ | $89.87_{\pm0.21}$ | $46.68_{\pm0.21}$ | $26.78_{\pm0.16}$ | $64.50_{\pm0.14}$ |

Table 6: Ablation study of target-domain finetuning by the 5-way 1-shot accuracy.

| Method | CropDisease | EuroSAT | ISIC2018 | ChestX | Ave. |
|---|---|---|---|---|---|
| Train CLS | $\mathbf{84.54}_{\pm0.23}$ | $\mathbf{74.97}_{\pm0.30}$ | $\mathbf{35.56}_{\pm0.17}$ | $\mathbf{23.39}_{\pm0.19}$ | $\mathbf{54.62}_{\pm0.13}$ |
| Fix CLS | $84.40_{\pm0.21}$ | $74.61_{\pm0.19}$ | $34.82_{\pm0.25}$ | $23.13_{\pm0.11}$ | $54.24_{\pm0.11}$ |
| Decouple CLS | $80.82_{\pm0.21}$ | $69.93_{\pm0.22}$ | $33.37_{\pm0.21}$ | $22.91_{\pm0.12}$ | $51.77_{\pm0.12}$ |

#### 4.3.2 Contribution of domain tokens

Subsequently, we study the contribution of domain tokens. We first directly remove the domain token in Tab. 5c (i.e., fixing the CLS token to random initialization, Tab. 1 a.2). We can see the performance is higher than the baseline model, but is much lower than our final method, which is consistent with Tab. 1 and verifies the domain token is crucial.

Then, we try to fix the domain token as random initialization (Tab. 5d). The performance is higher than Tab. 5c, indicating the separated domain information can indeed help to absorb domain information. Moreover, such absorption is enhanced by setting the domain token to learnable parameters as ours.

Finally, we try to randomize the domain choice in Tab. 5e, i.e., randomly choosing domain tokens for each sample. We can see the performance is even lower than Tab. 5c, indicating the domain token can indeed represent each pseudo domain and is important for CLS decoupling.

#### 4.3.3 Verification of target-domain finetuning

To study the contribution to target-domain finetuning, we report Tab. 6 for finetuning-based classifications under the most challenging 1-shot scenario. Compared with the default choice of finetuning (Fix CLS), our method (Train CLS) can indeed improve the finetuning performance, verifying the effectiveness of the CLS token in fast learning the domain information.

Moreover, we try to apply the CLS decoupling method to the finetuning, and the performance greatly drops. This is because there is no domain gap between the target-domain finetuning and evaluation, and the domain information is beneficial now. Therefore, the decoupling of domain information harms the performance, which on the contrary verifies the effectiveness of CLS decoupling.

### 4.4 Verification of Domain Tokens and Hyper-parameters

To study what the domain token encodes, we follow Sec. 2.3 to plot the similarity map between the domain token and image tokens in the corresponding class in Fig. 5. We can see the domain token can better capture the low-frequency regions (majorly represented as background) in each image compared with the baseline's CLS token (denoted as BL-CLS). Moreover, we quantitatively validated the domain distance in Fig. 2a, where we can see the domain similarity significantly increases by applying the domain token. These results indicate the domain can better absorb the domain information, so that it can better help to decouple such information from the CLS token.

Finally, we study the hyper-parameters in Fig. 5b,c. We can see only the CLS token in the first block (input layer) could be decoupled, consistent with Tab. 2. The best choice of cluster number is the number of source-domain classes, which means each class can be viewed as a pseudo domain. Since the source dataset (miniImageNet) is a general classification dataset, the difference between each class is larger (e.g., than fine-grained datasets where domain information is clear). Therefore, for

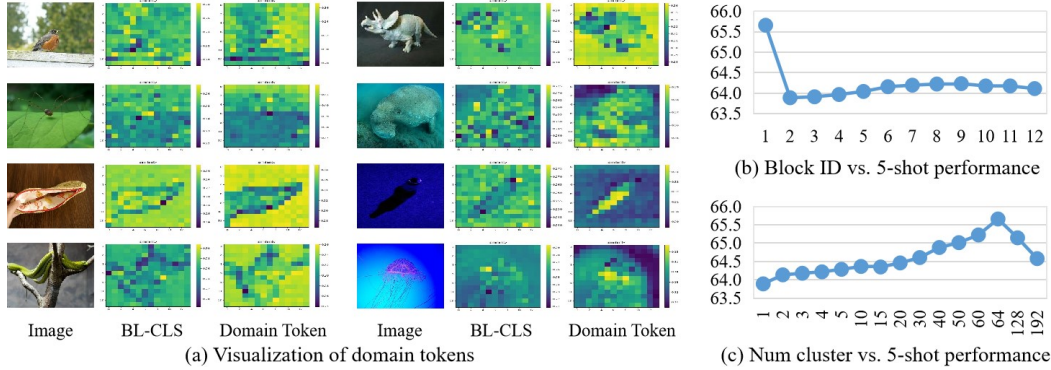

(a) Visualization of domain tokens

(b) Block ID vs. 5-shot performance

(c) Num cluster vs. 5-shot performance

Figure 5: (a) Domain tokens can better represent the background regions than the CLS token in the baseline method (BL-CLS), validating the effectiveness in absorbing domain information. (b) Applying the decoupling only on the first block (input layer) can improve the performance, consistent with Tab. 2. (c) The best pseudo-domain choice is to view each source-domain class as a domain.

miniImageNet, it is reasonable to view each class as a domain. Further experiments in the appendix verify that domain tokens absorb domain information instead of class-specific information.

## 5 Related Work

**Cross-Domain Few-Shot Learning (CDFSL)** has been studied by several works [14, 21, 28, 33, 47], which focuses on training a model on the source domain that can generalize well to target domain with limited examples. Current methods can be categorized into two types: meta-learning based approaches [12, 14, 17, 36], which aim at learning task-agnostic knowledge in order to learn new tasks efficiently, and transfer learning based approaches [4, 14, 20, 44, 46],tackling the problem based on reusing the model trained on the base classes data in a standard supervised learning way. However, these works are mostly restricted to the CNN architecture. Recently some works [13, 29, 35, 39] focus on the transformer structure to solve the CDFSL tasks but these efforts have not fully dug out the potential of the VIT structure and the importance of the CLS token on CDFSL.

**The CLS token.** Adding extra special tokens to the input sequence in the transformer architecture is popularized in BERT [9]. However, most methods extend the input tokens with new tokens in order to use their output value as the output of the model, like ViT adding a token as the CLS token[10]. Recently, some works [2, 23, 24] focus on using the CLS tokens in ViT to prune the network structure to reduce computational efficiency, and some researches [6] focus on adding additional tokens to the ViT to store and retrieve information. Different from them, our study is the first, as we know, to delve into the influence of the CLS token on cross-domain transferability, alleviating the domain gap by decoupling the CLS tokens into domain-specific and domain-agnostic tokens.

## 6 Conclusion

In this paper, we find a phenomenon that not loading the pretrained CLS token parameters can improve the CDFSL performance. We delve into this phenomenon for an interpretation, and find the CLS token naturally absorbs domain information due to ViT's structure. Based on the interpretation, we further propose a method to decouple the domain information in the CLS token for the CDFSL task. Extensive experiments on four CDFSL benchmarks validate our rationale and effectiveness.

## Acknowledgments

This work is supported by the National Natural Science Foundation of China under grants 62206102, 62376103, 62436003 and 62302184; the Science and Technology Support Program of Hubei Province under grant 2022BAA046; Hubei Science and Technology Talent Service Project under grant 2024DJC078; and Ant Group through CCF-Ant Research Fund.

## Footnotes

[2]Further experiments are conducted in the appendix to show not all tokens tend to have such a characteristic.

# References

[1] Mathilde Caron, Hugo Touvron, Ishan Misra, Hervé Jégou, Julien Mairal, Piotr Bojanowski, and Armand Joulin. Emerging properties in self-supervised vision transformers. In *Proceedings of the International Conference on Computer Vision (ICCV)*, 2021.

[2] Shuning Chang, Pichao Wang, Ming Lin, Fan Wang, David Junhao Zhang, Rong Jin, and Mike Zheng Shou. Making vision transformers efficient from a token sparsification view. In *Proceedings of the IEEE/CVF Conference on Computer Vision and Pattern Recognition*, pages 6195–6205, 2023.

[3] Xiangyu Chen, Qinghao Hu, Kaidong Li, Cuncong Zhong, and Guanghui Wang. Accumulated trivial attention matters in vision transformers on small datasets. In *Proceedings of the IEEE/CVF Winter Conference on Applications of Computer Vision (WACV)*, pages 3984–3992, January 2023.

[4] Xinyang Chen, Sinan Wang, Mingsheng Long, and Jianmin Wang. Transferability vs. discriminability: Batch spectral penalization for adversarial domain adaptation. In *Proceedings of the International Conference on Machine Learning*, pages 1081–1090. PMLR, 2019.

[5] Noel Codella, Veronica Rotemberg, Philipp Tschandl, M. Emre Celebi, Stephen Dusza, David Gutman, Brian Helba, Aadi Kalloo, Konstantinos Liopyris, Michael Marchetti, Harald Kittler, and Allan Halpern. Skin lesion analysis toward melanoma detection 2018: A challenge hosted by the international skin imaging collaboration (isic), 2019.

[6] Timothée Darcet, Maxime Oquab, Julien Mairal, and Piotr Bojanowski. Vision transformers need registers. In *The Twelfth International Conference on Learning Representations*, 2024.

[7] MohammadReza Davari, Stefan Horoi, Amine Natik, Guillaume Lajoie, Guy Wolf, and Eugene Belilovsky. Reliability of cka as a similarity measure in deep learning, 2022.

[8] Jia Deng, Wei Dong, Richard Socher, Li-Jia Li, Kai Li, and Li Fei-Fei. Imagenet: A large-scale hierarchical image database. In *Proceedings of the IEEE/CVF Conference on Computer Vision and Pattern Recognition*, pages 248–255. Ieee, 2009.

[9] Jacob Devlin, Ming-Wei Chang, Kenton Lee, and Kristina Toutanova. Bert: Pre-training of deep bidirectional transformers for language understanding. In *Proceedings of the Conference of the North American Chapter of the Association for Computational Linguistics: Human Language Technologies*, pages 4171–4186, 2019.

[10] Alexey Dosovitskiy, Lucas Beyer, Alexander Kolesnikov, Dirk Weissenborn, Xiaohua Zhai, Thomas Unterthiner, Mostafa Dehghani, Matthias Minderer, Georg Heigold, Sylvain Gelly, Jakob Uszkoreit, and Neil Houlsby. An image is worth 16x16 words: Transformers for image recognition at scale, 2021.

[11] Haoqi Fan, Bo Xiong, Karttikeya Mangalam, Yanghao Li, Zhicheng Yan, Jitendra Malik, and Christoph Feichtenhofer. Multiscale vision transformers. In *Proceedings of the IEEE/CVF International Conference on Computer Vision (ICCV)*, pages 6824–6835, October 2021.

[12] Yuqian Fu, Yu Xie, Yanwei Fu, Jingjing Chen, and Yu-Gang Jiang. Wave-san: Wavelet based style augmentation network for cross-domain few-shot learning, 2022.

[13] Yuqian Fu, Yu Xie, Yanwei Fu, and Yu-Gang Jiang. Styleadv: Meta style adversarial training for cross-domain few-shot learning, 2023.

[14] Yunhui Guo, Noel C Codella, Leonid Karlinsky, James V Codella, John R Smith, Kate Saenko, Tajana Rosing, and Rogerio Feris. A broader study of cross-domain few-shot learning. In *Proceedings of the IEEE/CVF European Conference on Computer Vision*, pages 124–141. Springer, 2020.

[15] Zixian Guo, Bowen Dong, Zhilong Ji, Jinfeng Bai, Yiwen Guo, and Wangmeng Zuo. Texts as images in prompt tuning for multi-label image recognition. In *Proceedings of the IEEE/CVF Conference on Computer Vision and Pattern Recognition*, pages 2808–2817, 2023.

[16] Patrick Helber, Benjamin Bischke, Andreas Dengel, and Damian Borth. Eurosat: A novel dataset and deep learning benchmark for land use and land cover classification, 2019.

[17] Yanxu Hu and Andy J. Ma. Adversarial feature augmentation for cross-domain few-shot classification, 2022.

[18] Seung Hoon Lee, Seunghyun Lee, and Byung Cheol Song. Vision transformer for small-size datasets. *arXiv preprint arXiv:2112.13492*, 2021.

[19] Pan Li, Shaogang Gong, Chengjie Wang, and Yanwei Fu. Ranking distance calibration for cross-domain few-shot learning, 2022.

[20] Hanwen Liang, Qiong Zhang, Peng Dai, and Juwei Lu. Boosting the generalization capability in cross-domain few-shot learning via noise-enhanced supervised autoencoder, 2021.

[21] Bingyu Liu, Zhen Zhao, Zhenpeng Li, Jianan Jiang, Yuhong Guo, and Jieping Ye. Feature transformation ensemble model with batch spectral regularization for cross-domain few-shot classification. *arXiv preprint arXiv:2005.08463*, 2020.

[22] Yicong Liu, Yixiong Zou, Ruixuan Li, and Yuhua Li. Spectral decomposition and transformation for cross-domain few-shot learning. *Neural Networks*, 179:106536, 2024.

[23] Sifan Long, Zhen Zhao, Jimin Pi, Shengsheng Wang, and Jingdong Wang. Beyond attentive tokens: Incorporating token importance and diversity for efficient vision transformers. In *Proceedings of the IEEE/CVF Conference on Computer Vision and Pattern Recognition*, pages 10334–10343, 2023.

[24] Chenyang Lu, Daan de Geus, and Gijs Dubbelman. Content-aware token sharing for efficient semantic segmentation with vision transformers. In *Proceedings of the IEEE/CVF Conference on Computer Vision and Pattern Recognition*, pages 23631–23640, 2023.

[25] Sharada P. Mohanty, David P. Hughes, and Marcel Salathé. Using deep learning for image-based plant disease detection. *Frontiers in Plant Science*, 7(September), September 2016. Publisher Copyright: © 2016 Mohanty, Hughes and Salathé.

[26] Muhammad Muzammal Naseer, Kanchana Ranasinghe, Salman H Khan, Munawar Hayat, Fahad Shahbaz Khan, and Ming-Hsuan Yang. Intriguing properties of vision transformers. In M. Ranzato, A. Beygelzimer, Y. Dauphin, P.S. Liang, and J. Wortman Vaughan, editors, *Advances in Neural Information Processing Systems*, volume 34, pages 23296–23308. Curran Associates, Inc., 2021.

[27] Mehrdad Noori, Milad Cheraghalikhani, Ali Bahri, Gustavo A. Vargas Hakim, David Osowiechi, Ismail Ben Ayed, and Christian Desrosiers. Tfs-vit: Token-level feature stylization for domain generalization. *Pattern Recognition*, 149:110213, 2024.

[28] Jaehoon Oh, Sungnyun Kim, Namgyu Ho, Jin-Hwa Kim, Hwanjun Song, and Se-Young Yun. Understanding cross-domain few-shot learning based on domain similarity and few-shot difficulty, 2022.

[29] Jan Stühmer Shell Xu, Da Li. Pushing the limits of simple pipelines for few-shot learning: External data and fine-tuning make a difference, 2022.

[30] Jake Snell, Kevin Swersky, and Richard Zemel. Prototypical networks for few-shot learning. In *Proceedings of the International Conference on Neural Information Processing Systems*, pages 4080–4090, 2017.

[31] Kihyuk Sohn, Huiwen Chang, José Lezama, Luisa Polania, Han Zhang, Yuan Hao, Irfan Essa, and Lu Jiang. Visual prompt tuning for generative transfer learning. In *Proceedings of the IEEE/CVF Conference on Computer Vision and Pattern Recognition*, pages 19840–19851, 2023.

[32] Maryam Sultana, Muzammal Naseer, Muhammad Haris Khan, Salman Khan, and Fahad Shahbaz Khan. Self-distilled vision transformer for domain generalization. In *Proceedings of the Asian Conference on Computer Vision (ACCV)*, pages 3068–3085, December 2022.

[33] Hung-Yu Tseng, Hsin-Ying Lee, Jia-Bin Huang, and Ming-Hsuan Yang. Cross-domain few-shot classification via learned feature-wise transformation. In *Proceedings of the International Conference on Learning Representations*, 2020.

[34] Oriol Vinyals, Charles Blundell, Timothy Lillicrap, Koray Kavukcuoglu, and Daan Wierstra. Matching networks for one shot learning. In *Proceedings of the International Conference on Neural Information Processing Systems*, pages 3637–3645, 2016.

[35] Reece Walsh, Islam Osman, and Mohamed S. Shehata. Masked embedding modeling with rapid domain adjustment for few-shot image classification. *IEEE Transactions on Image Processing*, 32:4907–4920, 2023.

[36] Haoqing Wang and Zhi-Hong Deng. Cross-domain few-shot classification via adversarial task augmentation, 2021.

[37] Jinpeng Wang, Pan Zhou, Mike Zheng Shou, and Shuicheng Yan. Position-guided text prompt for vision-language pre-training. In *Proceedings of the IEEE/CVF Conference on Computer Vision and Pattern Recognition*, pages 23242–23251, 2023.

[38] Xiaosong Wang, Yifan Peng, Le Lu, Zhiyong Lu, Mohammadhadi Bagheri, and Ronald M. Summers. Chestx-ray8: Hospital-scale chest x-ray database and benchmarks on weakly-supervised classification and localization of common thorax diseases. In *2017 IEEE Conference on Computer Vision and Pattern Recognition (CVPR)*. IEEE, July 2017.

[39] Kan Wu, Jinnian Zhang, Houwen Peng, Mengchen Liu, Bin Xiao, Jianlong Fu, and Lu Yuan. Tinyvit: Fast pretraining distillation for small vision transformers. In *European conference on computer vision (ECCV)*, 2022.

[40] Hantao Yao, Rui Zhang, and Changsheng Xu. Visual-language prompt tuning with knowledge-guided context optimization. In *Proceedings of the IEEE/CVF Conference on Computer Vision and Pattern Recognition*, pages 6757–6767, 2023.

[41] Chongzhi Zhang, Mingyuan Zhang, Shanghang Zhang, Daisheng Jin, Qiang Zhou, Zhongang Cai, Haiyu Zhao, Xianglong Liu, and Ziwei Liu. Delving deep into the generalization of vision transformers under distribution shifts. In *2022 IEEE/CVF Conference on Computer Vision and Pattern Recognition (CVPR)*, pages 7267–7276, 2022.

[42] Hao Zhang, Feng Li, Shilong Liu, Lei Zhang, Hang Su, Jun Zhu, Lionel M. Ni, and Heung-Yeung Shum. Dino: Detr with improved denoising anchor boxes for end-to-end object detection, 2022.

[43] Zhenyu Zhang, Guangyao Chen, Yixiong Zou, Zhimeng Huang, Yuhua Li, and Ruixuan Li. Micm: Rethinking unsupervised pretraining for enhanced few-shot learning. In *Proceedings of the 32nd ACM International Conference on Multimedia*, MM '24, page 7686–7695, New York, NY, USA, 2024. Association for Computing Machinery.

[44] Fei Zhou, Peng Wang, Lei Zhang, Wei Wei, and Yanning Zhang. Revisiting prototypical network for cross domain few-shot learning. In *Proceedings of the IEEE/CVF Conference on Computer Vision and Pattern Recognition (CVPR)*, pages 20061–20070, June 2023.

[45] Jinghao Zhou, Chen Wei, Huiyu Wang, Wei Shen, Cihang Xie, Alan Yuille, and Tao Kong. ibot: Image bert pre-training with online tokenizer. *arXiv preprint arXiv:2111.07832*, 2021.

[46] Yixiong Zou, Yicong Liu, Yiman Hu, Yuhua Li, and Ruixuan Li. Flatten long-range loss landscapes for cross-domain few-shot learning. In *Proceedings of the IEEE/CVF Conference on Computer Vision and Pattern Recognition (CVPR)*, pages 23575–23584, June 2024.

[47] Yixiong Zou, Shanghang Zhang, Jianpeng Yu, Yonghong Tian, and José MF Moura. Revisiting mid-level patterns for cross-domain few-shot recognition. In *Proceedings of the ACM International Conference on Multimedia*, pages 741–749, 2021.

# A    Appendix / supplemental material

## A.1    Detailed Dataset Description

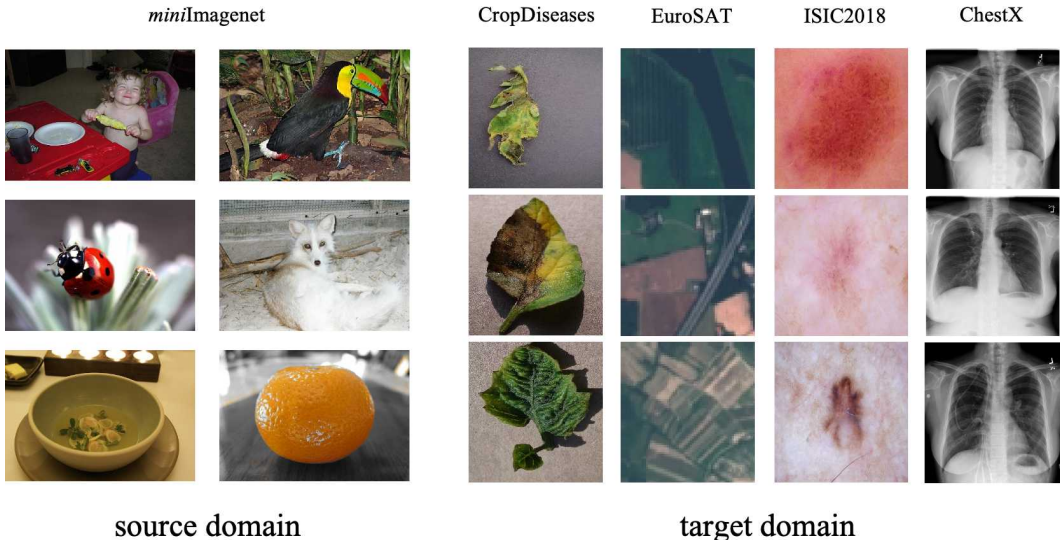

Figure 6: Samples of source domain *mini*ImageNet dataset(left) and target domain datasets (right), from left to right correspond to five distinct datasets: *mini*ImageNet, CropDiseases, EuroSAT, ISIC2018, and ChestX.

***mini*ImageNet**[34] is a subset of the ImageNet[8] dataset that contains 100 categories, each consisting of 600 natural images. Following the current work[28, 33], we split the *mini*ImageNet into 64 classes as the source domain training dataset. In addition, as shown in Figure 6, we utilize the datasets from four other different domains, like agriculture, remote sensing, and medical data, as target domains. We'll sequentially introduce each of them below.

**CropDiseases** [25] consists of 38 distinct classes and a total of 43,456 images, which are natural images, but are very specialized (specific to the agriculture industry), including various infected crops, healthy plants, and their corresponding disease category labels.

**EuroSAT** [16] contains a total of 27,000 satellite images of the Earth categorized into 10 distinct classes. The images in the EuroSAT are less similar to images in *mini*ImageNet since they lack perspective distortion, but still color images of natural scenes.

**ISIC2018** [5], which is even less similar to the *mini*ImageNet as they could not even represent natural scenes, encompasses 10,015 medical images for skin lesion classification across 7 different classes.

**ChestX** [38], a medical dataset for chest classification, consists of 25,847 images distributed across 7 distinct classes. The dataset is the most dissimilar to the *mini*ImageNet in three criteria. Apart from the two factors mentioned above, it loses 2 color channels that appear in the ChestX.

## A.2    More Experiments

### A.2.1    More Visualization of the CLS token by retrieval

To delve into the information encoded by the CLS token, we calculate the cosine similarity between the CLS token and the image token in the input layer of a fixed ViT trained on the source dataset and report the similarity map in Figure 7. It is clear that on the source dataset, the background regions have a higher similarity map, roughly representing the contour of the object for the source domain dataset. It means that the CLS token can distinguish the background and the foreground of the image which could indeed facilitate source-domain recognition. However, as shown in Figure 7 from above to bottom, with images that are less similar to the source domain, the ability to recognize the background for the CLS token gradually fades into mediocrity.

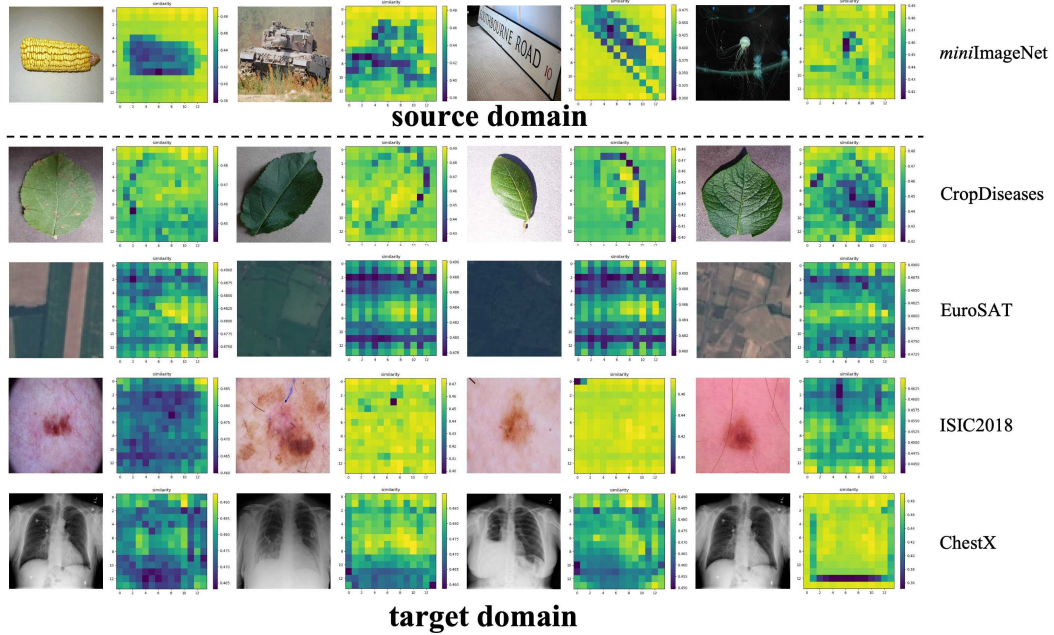

Figure 7: Samples from top to bottom correspond to the similarity map between the CLS token and the image token on the *mini*ImageNet and four target domain datasets, like CropDiseases, EuroSAT, ISIC2018, and ChestX. It can be seen that the CLS token can distinguish the background and the foreground of the image which could indeed facilitate the source-domain recognition on the source dataset while meeting more difficulties with a larger domain gap.

### A.2.2 More Visualization of the domain tokens by retrieval

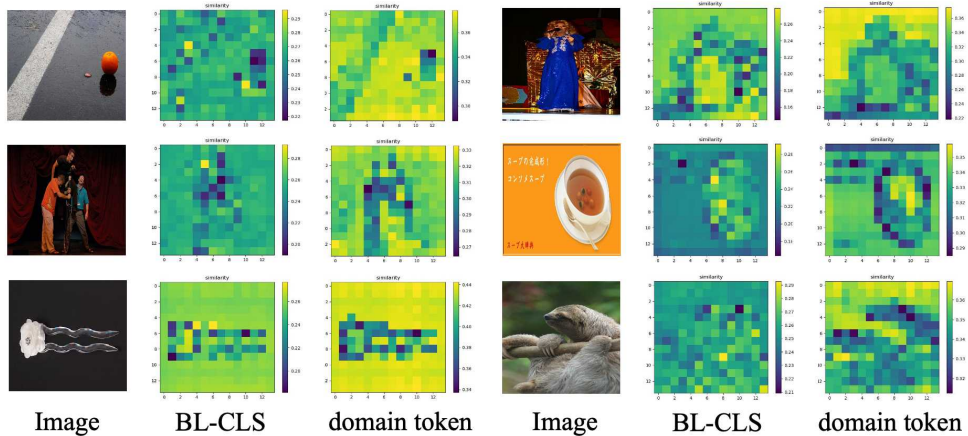

| Image | BL-CLS | domain token | Image | BL-CLS | domain token |

Figure 8: Applying the domain token significantly improves the domain similarity compared to the CLS token of the baseline method (BL-CLS), validating the effectiveness of our approach in absorbing domain information

To delve into what the domain token encodes, we re-utilize the similarity map plotted in Figure 8 as our measurement to compare our approach with the baseline's CLS token (denoted as BL-CLS). It seems that when utilizing the domain token, the domain similarity significantly increases, proving

that the domain token can better absorb the domain information and efficiently help to decouple such information from the CLS token.

### A.2.3 Applying Our Method to Other Baselines

Table 7: Ablation study of our method with iBOT-pretrained ViT and DINO-Pretrained ViT-Base by the 5-way 1-shot accuracy.

| Method | CropDiseases | EuroSAT | ISIC2018 | ChestX | Ave. |
|---|---|---|---|---|---|
| iBOT | 81.17 | 72.71 | 31.44 | 22.56 | 51.97 |
| **iBOT + Ours** | **81.31** | **72.80** | **31.87** | **22.57** | **52.14** |
| DINO-ViT-Base | 82.97 | 72.06 | 34.19 | 22.60 | 52.95 |
| **DINO-ViT-Base + Ours** | **83.11** | **73.77** | **34.75** | **22.98** | **53.65** |

Table 8: Implementing our method with meta-learning baseline.

| Method | Crop. | Euro. | ISIC. | Ches. | Ave. |
|---|---|---|---|---|---|
| ProtoNet | 93.59 | 86.92 | 46.15 | 25.68 | 63.09 |
| **ProtoNet + Ours** | **95.03** | **89.42** | **48.67** | **27.15** | **65.07** |

We also implement our approach on distinct backbones, like ViT pretrained by iBOT [45], and ViT-Base [42] pretrained by DINO. The results can be seen in Tab 7. Specifically, iBOT represents the iBOT-pretrained ViT baseline, and DINO-ViT-Base corresponds to the ViT-Base pretrained by DINO baseline. It is clarified from the average performance of four target domains that our approach shows improvements among both backbones in the 5-way 1-shot setting.

To verify our model also suits the meta-learning-based baselines, we conduct experiments based on the ProtoNet [30] in Tab 8. We can see that our model also improves this kind of baseline method.

### A.2.4 Further Verification of the CLS Token

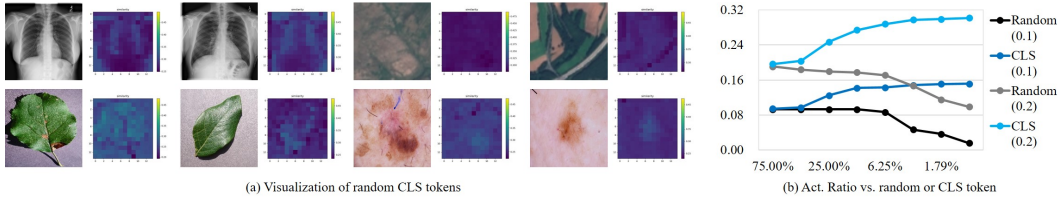

(a) Visualization of random CLS tokens    (b) Act. Ratio vs. random or CLS token

Figure 9: To verify it is the CLS token that tends to capture low-frequency components, we (a) visualize similarity maps of a random token and (b) use random tokens to calculate the similarity map of low-frequency images, and find random tokens do not show the same results as the CLS token.

To verify it is the CLS token that tends to capture low-frequency components, we visualize similarity maps of a random token in Fig. 9a. We use the same color bar as in Fig.2b. We can see the similarity is much lower, but a coarse contour of objects can still be observed in both the source and target domains, indicating a good transferability of detecting object contours, because no domain information is in the random token. However, the contour detected by the random token is much worse than that by the CLS tokens as shown in Fig.2a and Fig.5a, indicating although the random CLS token can initially detect the object contour, the learning of the CLS token strengthens this characteristic to detect low-frequency images.

Then, we use random tokens to calculate the similarity map of low-frequency images, and find random tokens do not show the same results as the CLS token. Specifically, we measure the activation ratio of the CLS token and the random token. We take the top 10% or 20% value as examples in Fig. 9b. We can see the random tokens show a tendency to decrease the activation ratio, while the CLS token shows a tendency to increase the ratio, indicating it is the CLS token that tends to be similar to the feature of low-frequency images.

Table 9: Verification of re-initializing the CLS token.

| Method | Crop. | Euro. | ISIC. | Ches. | Ave. |
|---|---|---|---|---|---|
| Baseline | 95.62 | 88.62 | 46.08 | 26.25 | 63.89 |
| Ours | 95.55 | 90.48 | 49.58 | 27.03 | 65.66 |
| Ours + Re-Init | 95.20 | 89.60 | 50.23 | 26.79 | 65.31 |

### A.2.5  Further Verification of Domain Tokens

To verify that our model does not overfit the fixed random CLS token (so that view the fixed value as a kind of new domain token), we re-initialize the CLS token during the source-domain training. The results are reported in Tab. 9, and we can see the improvements also exist.

However, re-initializing the CLS token can be viewed as adding noise to the domain token, therefore harming the absorbed domain information, which then affects the domain-irrelevant information learned by other structures in ViT. As a result, the Re-Init performance is slightly lower.

Indeed, domain tokens are encouraged to be orthogonal to the fixed CLS token, which would drive the model to view the fixed token as a domain-agnostic token. But note that the random token is already agnostic enough to every domain even without training, therefore our training would not essentially drive the model to be more agnostic to that CLS token, i.e., our model is not bound to the specific value of the CLS token.

Table 10: Training with datasets of 5 constructed domains.

| Method | Crop. | Euro. | ISIC. | Ches. | Ave. |
|---|---|---|---|---|---|
| Baseline | 93.85 | 89.72 | 49.74 | 26.07 | 64.77 |
| 320 classes as domain token | 95.03 | 90.49 | 49.10 | 26.81 | 65.35 |
| 64 classes as domain token | 95.16 | 90.61 | 47.86 | 26.77 | 65.10 |
| 5 domains as domain token | **95.52** | **90.62** | **50.77** | **27.00** | **65.98** |

To further ablate "domain-specific" and "class-specific", we then manually construct some new source domains based on miniImageNet. Specifically, we take the amplitude (by Fourier transformation) from target domains as the style information, and use the phase (by Fourier transformation) from the original source-domain images as the content information, thereby constructing 4 new domains with the original 64 source-domain classes. Then, we train our model on a new dataset containing the 4 constructed datasets and the original source dataset, and ablate different choice of domain tokens in Tab. 10.

As can be seen, by introducing larger domain gaps, viewing each class as a domain is not the best choice. Instead, setting a domain token for each domain could achieve the best performance, which validates the rationale of the domain token in absorbing the domain information.

### A.3  Broader Impact

We propose a CD-FSL method based on decoupling the CLS token into domain-specific and domain-agnostic tokens in the ViT and making use of it for efficient downstream few-shot learning to alleviate the domain gap and generalize well to the target domain. This work can also be adopted in other fields, like domain generalization, domain adaption, and few-shot class-incremental learning, where the challenge of enhancing the transferability of model exists universally. The evaluations of our approach are mainly across four different target domains, which may not represent all possible real-world scenarios. The approach can be evaluated on various target domains to be validated in a more realistic setting.

